# Boosting Algorithms for Maximizing the Soft Margin

**Manfred K. Warmuth**[*]
Dept. of Engineering
University of California
Santa Cruz, CA, U.S.A.

**Karen Glocer**
Dept. of Engineering
University of California
Santa Cruz, CA, U.S.A.

**Gunnar Rätsch**
Friedrich Miescher Laboratory
Max Planck Society
Tübingen, Germany

## Abstract

We present a novel boosting algorithm, called SoftBoost, designed for sets of binary labeled examples that are not necessarily separable by convex combinations of base hypotheses. Our algorithm achieves robustness by *capping* the distributions on the examples. Our update of the distribution is motivated by minimizing a relative entropy subject to the capping constraints and constraints on the *edges* of the obtained base hypotheses. The capping constraints imply a *soft margin* in the dual optimization problem. Our algorithm produces a convex combination of hypotheses whose soft margin is within $\delta$ of its maximum. We employ relative entropy projection methods to prove an $\mathcal{O}(\frac{\ln N}{\delta^2})$ iteration bound for our algorithm, where $N$ is number of examples.

We compare our algorithm with other approaches including LPBoost, BrownBoost, and SmoothBoost. We show that there exist cases where the number of iterations required by LPBoost grows linearly in $N$ instead of the logarithmic growth for SoftBoost. In simulation studies we show that our algorithm converges about as fast as LPBoost, faster than BrownBoost, and much faster than SmoothBoost. In a benchmark comparison we illustrate the competitiveness of our approach.

## 1 Introduction

Boosting methods have been used with great success in many applications like OCR, text classification, natural language processing, drug discovery, and computational biology [13]. For AdaBoost [7] it was frequently observed that the generalization error of the combined hypotheses kept decreasing after the training error had already reached zero [19]. This sparked a series of theoretical studies trying to understand the underlying principles that govern the behavior of ensemble methods [19, 1]. It became apparent that some of the power of ensemble methods lies in the fact that they tend to increase the margin of the training examples. This was consistent with the observation that AdaBoost works well on low-noise problems, such as digit recognition tasks, but not as well on tasks with high noise. On such tasks, better generalizaton can be achieved by not enforcing a large margin on *all* training points. This experimental observation was supported by the study of [19], where the generalization error of ensemble methods was bounded by the sum of two terms: the fraction of training points which have a margin smaller than some value $\rho$ plus a complexity term that depends on the base hypothesis class and $\rho$. While this worst-case bound can only capture part of what is going on in practice, it nevertheless suggests that in some cases it pays to allow some points to have small margin or be misclassified if this leads to a larger overall margin on the remaining points.

To cope with this problem, it was necessary to construct variants of AdaBoost which trade off the fraction of examples with margin at least $\rho$ with the size of the margin $\rho$. This was typically done by preventing the distribution maintained by the algorithm from concentrating too much on the most difficult examples. This idea is implemented in many algorithms including AdaBoost with soft margins [15], MadaBoost [5], $\nu$-Arc [16, 14], SmoothBoost [21], LPBoost [4], and several others (see references in [13]). For some of these algorithms, significant improvements were shown compared to the original AdaBoost algorithm on high noise data.

---

[*]Supported by NSF grant CCR 9821087.

In parallel, there has been a significant interest in how the linear combination of hypotheses generated by AdaBoost is related to the maximum margin solution [1, 19, 4, 18, 17]. It was shown that AdaBoost generates a combined hypothesis with a large margin, but not necessarily the maximum hard margin [15, 18]. This observation motivated the development of many Boosting algorithms that aim to maximize the margin [1, 8, 4, 17, 22, 18]. AdaBoost$^*$ [17] and TotalBoost [22] provable converge to the maximum hard margin within precision $\delta$ in $2\ln(N/\delta^2)$ iterations. The other algorithms have worse or no known convergence rates. However, such margin-maximizing algorithms are of limited interest for a practitioner working with noisy real-world data sets, as overfitting is even more problematic for such algorithms than for the original AdaBoost algorithm [1, 8].

In this work we combine these two lines of research into a single algorithm, called SoftBoost, that for the first time implements the soft margin idea in a practical boosting algorithm. SoftBoost finds in $O(\ln(N)/\delta^2)$ iterations a linear combination of base hypotheses whose soft margin is at least the optimum soft margin minus $\delta$. BrownBoost [6] does not always optimize the soft margin. SmoothBoost and MadaBoost can be related to maximizing the soft margin, but while they have known iterations bounds in terms of other criteria, it is unknown how quickly they converge to the maximum soft margin. From a theoretical point of view the optimization problems underlying SoftBoost as well as LPBoost are appealing, since they directly maximize the margin of a (typically large) *subset* of the training data [16]. This quantity plays a crucial role in the generalization error bounds [19].

Our new algorithm is most similar to LPBoost because its goal is also to optimize the soft margin. The most important difference is that we use slightly relaxed constraints and a relative entropy to the uniform distribution as the objective function. This leads to a distribution on the examples that is closer to the uniform distribution. An important result of our work is to show that this strategy may help to increase the convergence speed: We will give examples where LPBoost converges much more slowly than our algorithm—linear versus logarithmic growth in $N$.

The paper is organized as follows: in Section 2 we introduce the notation and the basic optimization problem. In Section 3 we discuss LPBoost and give a separable setting where $N/2$ iterations are needed by LPBoost to achieve a hard margin within precision .99. In Section 4 we present our new SoftBoost algorithm and prove its iteration bound. We provide an experimental comparison of the algorithms on real and synthetic data in Section 5, and conclude with a discussion in Section 6.

## 2 Preliminaries

In the boosting setting, we are given a set of $N$ labeled training examples $(x_n, y_n)$, $n = 1 \ldots N$, where the instances $x_n$ are in some domain $\mathcal{X}$ and the labels $y_n \in \pm 1$. Boosting algorithms maintain a distribution $\mathbf{d}$ on the $N$ examples, i.e. $\mathbf{d}$ lies in the $N$ dimensional probability simplex $\mathcal{P}^N$. Intuitively, the hard to classify examples receive more weight. In each iteration, the algorithm gives the current distribution to an oracle (a.k.a. base learning algorithm), which returns a new base hypothesis $h : \mathcal{X} \to [-1, 1]^N$ with a certain guarantee of performance. This guarantee will be discussed at the end of this section.

One measure of the performance of a base hypothesis $h$ with respect to distribution $\mathbf{d}$ is its edge, $\gamma_h = \sum_{n=1}^{N} d_n y_n h(x_n)$. When the range of $h$ is $\pm 1$ instead of the interval [-1,1], then the edge is just an affine transformation of the weighted error $\epsilon_h$ of hypothesis $h$: i.e. $\epsilon_h(\mathbf{d}) = \frac{1}{2} - \frac{1}{2}\gamma_h$. A hypothesis that predicts perfectly has edge $\gamma = 1$, a hypothesis that always predicts incorrectly has edge $\gamma = -1$, and a random hypothesis has edge $\gamma \approx 0$. The higher the edge, the more useful is the hypothesis for classifying the training examples. The edge of a set of hypotheses is defined as the maximum edge of the set.

After a hypothesis is received, the algorithm must update its distribution $\mathbf{d}$ on the examples. Boosting algorithms (for the separable case) commonly update their distribution by placing a constraint on the edge of most recent hypothesis. Such algorithms are called *corrective* [17]. In *totally corrective* updates, one constrains the distribution to have small edge with respect to *all* of the previous hypotheses [11, 22]. The update developed in this paper is an adaptation of the totally corrective update of [22] that handles the inseparable case. The final output of the boosting algorithm is always a convex combination of base hypotheses $f_{\mathbf{w}}(x_n) = \sum_{t=1}^{T} w_t h^t(x_n)$, where $h^t$ is the hypothesis added at iteration $t$ and $w_t$ is its coefficient. The margin of a labeled example $(x_n, y_n)$ is defined as

$\rho_n = y_n f_{\mathbf{w}}(x_n)$. The (hard) margin of a set of examples is taken to be the minimum margin of the set.

It is convenient to define an $N$-dimensional vector $\mathbf{u}^m$ that combines the base hypothesis $h^m$ with the labels $y_n$ of the $N$ examples: $u_n^m := y_n h^m(x_n)$. With this notation, the edge of the $t$-th hypothesis becomes $\mathbf{d} \cdot \mathbf{u}^t$ and the margin of the $n$-th example w.r.t. a convex combination $\mathbf{w}$ of the first $t-1$ hypotheses is $\sum_{m=1}^{t-1} u_n^m w_m$.

For a given set of hypotheses $\{h^1, \ldots, h^t\}$, the following linear programming problem (1) optimizes the minimum *soft margin*. The term "soft" here refers to a relaxation of the margin constraint. We now allow examples to lie below the margin but penalize them linearly via slack variables $\psi_n$. The dual problem (2) minimizes the maximum edge when the distribution is capped with $1/\nu$, where $\nu \in \{1, \ldots, N\}$:

$$\rho_t^*(\nu) = \max_{\mathbf{w}, \rho, \boldsymbol{\psi}} \left( \rho - \frac{1}{\nu} \sum_{n=1}^{N} \psi_n \right) \quad (1) \qquad\qquad \gamma_t^*(\nu) = \min_{\mathbf{d}, \gamma} \gamma \quad (2)$$

$$\text{s.t. } \sum_{m=1}^{t} u_n^m w_m \geq \rho - \psi_n, \text{ for } 1 \leq n \leq N, \qquad \text{s.t. } \mathbf{d} \cdot \mathbf{u}^m \leq \gamma, \text{ for } 1 \leq m \leq t,$$

$$\mathbf{w} \in \mathcal{P}^t, \ \boldsymbol{\psi} \geq \mathbf{0}. \qquad\qquad \mathbf{d} \in \mathcal{P}^N, \ \mathbf{d} \leq \frac{1}{\nu} \mathbf{1}.$$

By duality, $\rho_t^*(\nu) = \gamma_t^*(\nu)$. Note that the relationship between capping and the hinge loss has long been exploited by the SVM community [3, 20] and has also been used before for Boosting in [16, 14]. In particular, it is known that $\rho$ in (1) is chosen such that $N - \nu$ examples have margin at least $\rho$. This corresponds to $\nu$ active constraints in (2). The case $\nu = 1$ is degenerate: there are no capping constraints in (2) and this is equivalent to the hard margin case.[1]

**Assumption on the weak learner** We assume that for any distribution $\mathbf{d} \leq \frac{1}{\nu} \mathbf{1}$ on the examples, the oracle returns a hypothesis $h$ with edge at least $g$, for some fixed $g$. This means that for the corresponding $\mathbf{u}$ vector, $\mathbf{d} \cdot \mathbf{u} \geq g$. For binary valued features, this is equivalent to the assumption that the base learner always returns a hypothesis with error at most $\frac{1}{2} - \frac{1}{2} g$.

Adding a new constraint can only increase the value $\gamma_t^*(\nu)$ of the minimization problem (2) and therefore $\gamma_t^*(\nu)$ is non-decreasing in $t$. It is natural to define $\gamma^*(\nu)$ as the value of (2) w.r.t. the entire hypothesis set from which the oracle can choose. Clearly $\gamma_t^*(\nu)$ approaches $\gamma^*(\nu)$ from below. Also, the guarantee $g$ of the oracle can be at most $\gamma^*(\nu)$ because for the optimal distribution $\mathbf{d}^*$ that realizes $\gamma^*(\nu)$, all hypotheses have edge at most $\gamma^*(\nu)$. For computational reasons, $g$ might however be lower than $\gamma^*(\nu)$ and in that case the optimum soft margin we can achieve is $g$.

## 3 LPBoost

In iteration $t$, the LPBoost algorithm [4] sends its current distribution $\mathbf{d}^{t-1}$ to the oracle and receives a hypothesis $h^t$ that satisfies $\mathbf{d}^{t-1} \cdot \mathbf{u}^t \geq g$. It then updates its distribution to $\mathbf{d}^t$ by solving the linear programming problem (1) based on the $t$ hypotheses received so far.

The goal of the boosting algorithms is to produce a convex combination of $T$ hypotheses such that $\gamma_T(\nu) \geq g - \delta$. The simplest way to achieve this is to break when this condition is satisfied. Although the guarantee $g$ is typically not known, it is upper bounded by $\widehat{\gamma}_t = \min_{1 \leq m \leq t} \mathbf{d}^{t-1} \cdot \mathbf{u}^t$ and therefore LPBoost uses the more stringent stopping criterion $\gamma_t(\nu) \geq \widehat{\gamma}_t - \delta$.

To our knowledge, there is no known iteration bound for LPBoost even though it provably converges to the $\delta$-optimal solution of the optimization problem after it has seen all hypotheses [4, 10]. Empirically, the convergence speed depends on the linear programming optimizer, e.g. simplex or interior point solver [22]. For the first time, we are able to establish a lower bound showing that, independent of the optimizer, LPBoost can require $\Omega(N)$ iterations:

**Theorem 1** *There exists a case where LPBoost requires $N/2$ iterations to achieve a hard margin that is within $\delta = .99$ of the optimum hard margin.*
*Proof.* Assume we are in the hard margin case ($\nu = 1$). The counterexample has $N$ examples and $\frac{N}{2} + 1$ base hypothesis. After $\frac{N}{2}$ iterations, the optimal value $\gamma_t^*(1)$ for the chosen hypotheses will

**Algorithm 1** LPBoost with accuracy param. $\delta$ and capping parameter $\nu$

1. **Input:** $S = \langle (x_1, y_1), \ldots, (x_N, y_N) \rangle$, accuracy $\delta$, capping parameter $\nu \in [1, N]$.
2. **Initialize:** $\mathbf{d}^0$ to the uniform distribution and $\widehat{\gamma}_0$ to 1.
3. **Do for** $t = 1, \ldots$
   (a) Send $\mathbf{d}^{t-1}$ to oracle and obtain hypothesis $h^t$.
   Set $u_n^t = h^t(x_n) y_n$ and $\widehat{\gamma}_t = \min\{\widehat{\gamma}_{t-1}, \mathbf{d}^{t-1} \cdot \mathbf{u}^t\}$.
   (Assume $\mathbf{d}^{t-1} \cdot \mathbf{u}^t \geq g$, where edge guarantee $g$ is unknown.)

   (b) Update the distribution to any $\mathbf{d}^t$ that solves the LP problem
   $$(\mathbf{d}^t, \gamma_t^*) = \underset{\mathbf{d}, \gamma}{\operatorname{argmin}} \ \gamma \quad \text{s.t.} \quad \mathbf{d} \cdot \mathbf{u}^m \leq \gamma, \text{ for } 1 \leq m \leq t; \mathbf{d} \in \mathcal{P}^N, \mathbf{d} \leq \frac{1}{\nu}\mathbf{1}.$$
   (c) **If** $\gamma_t^* \geq \widehat{\gamma}_t - \delta$ **then** set $T = t$ and break.[2]

4. **Output:** $f_{\mathbf{w}}(x) = \sum_{m=1}^{T} w_m h^m(x)$, where the coefficients $\mathbf{w}_m$ maximize the soft margin over the hypothesis set $\{h^1, \ldots, h^T\}$ using the LP problem (1).

[2]When $g$ is known, then one can break already when $\gamma_t^*(\nu) \geq g - \delta$.

---

still be close to $-1$, whereas after the last hypothesis is added, this value is at least $\epsilon/2$. Here $\epsilon$ is a precision parameter that is an arbitrary small number.

Figure 1 shows the case where $N = 8$ and $T = 5$, but it is trivial to generalize this example to any even $N$. There are 8 examples/rows and the five columns are the $\mathbf{u}^t$'s of the five available base hypotheses. The examples are separable because if we put half of the weight on the first and last hypothesis, then the margins of all examples are at least $\epsilon/2$.

| $n \backslash t$ | 1 | 2 | 3 | 4 | 5 |
|---|---|---|---|---|---|
| 1 | $+1$ | $-1 + 5\epsilon$ | $-1 + 7\epsilon$ | $-1 + 9\epsilon$ | $-1 + \epsilon$ |
| 2 | $+1$ | $-1 + 5\epsilon$ | $-1 + 7\epsilon$ | $-1 + 9\epsilon$ | $-1 + \epsilon$ |
| 3 | $+1$ | $-1 + 5\epsilon$ | $-1 + 7\epsilon$ | $-1 + 9\epsilon$ | $-1 + \epsilon$ |
| 4 | $+1$ | $-1 + 5\epsilon$ | $-1 + 7\epsilon$ | $-1 + 9\epsilon$ | $-1 + \epsilon$ |
| 5 | $\mathbf{-1 + 2\epsilon}$ | $+1$ | $-1 + 7\epsilon$ | $-1 + 9\epsilon$ | $+1 - \epsilon$ |
| 6 | $-1 + 3\epsilon$ | $\mathbf{-1 + 4\epsilon}$ | $+1$ | $-1 + 9\epsilon$ | $+1 - \epsilon$ |
| 7 | $-1 + 3\epsilon$ | $-1 + 5\epsilon$ | $\mathbf{-1 + 6\epsilon}$ | $+1$ | $+1 - \epsilon$ |
| 8 | $-1 + 3\epsilon$ | $-1 + 5\epsilon$ | $-1 + 7\epsilon$ | $\mathbf{-1 + 8\epsilon}$ | $+1 - \epsilon$ |
| $\gamma_t^*(1)$ | $\mathbf{-1 + 2\epsilon}$ | $\mathbf{-1 + 4\epsilon}$ | $\mathbf{-1 + 6\epsilon}$ | $\mathbf{-1 + 8\epsilon}$ | $\geq \epsilon/2$ |

**Figure 1:** The $\mathbf{u}^t$ vectors that are hard for LPBoost (for $\nu = 1$).

We assume that in each iteration the oracle will return the remaining hypothesis with maximum edge. This will result in LPBoost choosing the hypotheses in order, and there will never be any ties. The initial distribution $\mathbf{d}^0$ is uniform. At the end of iteration $t$ ($1 \leq t \leq N/2$), the distribution $\mathbf{d}^t$ will focus all its weight on example $N/2 + t$, and the optimum mixture of the columns will put all of its weight on the $t$th hypothesis that was just received. In other words the value will be the bolded entries in Figure 1: $-1 + 2\epsilon t$ at the end of iteration $t = 1, \ldots, N/2$. After $N/2$ iterations the value $\gamma_t^*(1)$ of the underlying LP problem will still be close to $-1$, because $\epsilon$ can be made arbitrary small. We reasoned already that the value for all $N/2 + 1$ hypotheses will be positive. So if $\epsilon$ is small enough, then after $N/2$ iterations LPBoost is still at least .99 away from the optimal solution. $\square$

Although the example set used in the above proof is linearly separable, we can modify it explicitly to argue that capping the distribution on examples will not help in the sense that "soft" LPBoost with $\nu > 1$ can still have linear iteration bounds. To negate the effect of capping, simply pad out the problem by duplicating all of the rows $\nu$ times. There will now be $\tilde{N} = N\nu$ examples, and after $\frac{N}{2} = \frac{\tilde{N}}{2\nu}$ iterations, the value of the game is still close to $-1$. This is not a claim that capping has no value. It remains an important technique for making an algorithm more robust to noise. However, it is not sufficient to improve the iteration bound of LPBoost from linear growth in $N$ to logarithmic.

Another attempt might be to modify LPBoost so that at each iteration a base hypothesis is chosen that increases the value of the optimization problem the most. Unfortunately we found similar $\Omega(N)$ counter examples to this heuristic (not shown). It is also easy to see that the algorithms related to the below SoftBoost algorithm choose the last hypothesis after first and finish in just two iterations.

**Algorithm 2** SoftBoost with accuracy param. $\delta$ and capping parameter $\nu$

---

1. **Input:** $S = \langle (x_1, y_1), \ldots, (x_N, y_N) \rangle$, desired accuracy $\delta$, and capping parameter $\nu \in [1, N]$.

2. **Initialize:** $\mathbf{d}^0$ to the uniform distribution and $\widehat{\gamma}_0$ to 1.

3. **Do for** $t = 1, \ldots$

   (a) Send $\mathbf{d}^{t-1}$ to the oracle and obtain hypothesis $h^t$.
       Set $u_n^t = h^t(x_n)y_n$ and $\widehat{\gamma}_t = \min\{\widehat{\gamma}_{t-1}, \mathbf{d}^{t-1} \cdot \mathbf{u}^t\}$.
       (Assume $\mathbf{d}^{t-1} \cdot \mathbf{u}^t \geq g$, where edge guarantee $g$ is unknown.)

   (b) Update[3]
       $$\mathbf{d}^t = \underset{\mathbf{d}}{\operatorname{argmin}} \, \Delta(\mathbf{d}, \mathbf{d}^0), \quad \text{s.t. } \mathbf{d} \cdot \mathbf{u}^m \leq \widehat{\gamma}_t - \delta, \text{ for } 1 \leq m \leq t, \sum_n d_n = 1, \, \mathbf{d} \leq \frac{1}{\nu}\mathbf{1}.$$

   (c) **If** above infeasible or $\mathbf{d}^t$ contains a zero **then** $T = t$ and break.

4. **Output:** $f_{\mathbf{w}}(x) = \sum_{m=1}^T \mathbf{w}_m h^m(x)$, where the coefficients $\mathbf{w}_m$ maximize the soft margin over the hypothesis set $\{h^1, \ldots, h^t\}$ using the LP problem (1).

[3] When $g$ is known, replace the upper bound $\widehat{\gamma}_t - \delta$ by $g - \delta$.

## 4 SoftBoost

In this section, we present the SoftBoost algorithm, which adds capping to the TotalBoost algorithm of [22]. SoftBoost takes as input a sequence of examples $S = \langle (x_1, y_1), \ldots, (x_N, y_N) \rangle$, an accuracy parameter $\delta$, and a capping parameter $\nu$. The algorithm has an oracle available with unknown guarantee $g$. Its initial distribution $\mathbf{d}^0$ is uniform. In each iteration $t$, the algorithm prompts the oracle for a new base hypothesis, incorporates it into the constraint set, and updates its distribution $\mathbf{d}^{t-1}$ to $\mathbf{d}^t$ by minimizing the relative entropy $\Delta(\mathbf{d}, \mathbf{d}^0) := \sum_n d_n \ln \frac{d_n}{d_n^0}$ subject to linear constraints:

$$\mathbf{d}^{t+1} = \operatorname{argmin}_{\mathbf{d}} \Delta(\mathbf{d}, \mathbf{d}^0)$$
$$\text{s.t.} \quad \mathbf{d} \cdot \mathbf{u}^m \leq \widehat{\gamma}_t - \delta, \text{ for } 1 \leq m \leq t \text{ (where } \widehat{\gamma}_t = \min_{1 \leq m \leq t} \mathbf{d}^{m-1} \cdot \mathbf{u}^m),$$
$$\sum_n d_n = 1, \, \mathbf{d} \leq \frac{1}{\nu}\mathbf{1}.$$

It is easy to solve this optimization problem with vanilla sequential quadratic programming methods (see [22] for details). Observe that removing the relative entropy term from the objective, results in a feasibility problem for linear programming where the edges are upper bounded by $\widehat{\gamma}_t - \delta$. If we remove the relative entropy and minimize the upper bound on the edges, then we arrive at the optimization problem of LPBoost, and logarithmic growth in the number of examples is no longer possible. The relative entropy in the objective assures that the probabilities of the examples are always proportional to their exponentiated negative soft margins (not shown). That is, more weight is put on the examples with low soft margin, which are the examples that are hard to classify.

### 4.1 Iteration bounds for SoftBoost

Our iteration bound for SoftBoost is very similar to the bound proven for TotalBoost [22], differing only in the additional details related to capping.

**Theorem 2** *SoftBoost terminates after at most $\lceil \frac{2}{\delta^2} \ln(N/\nu) \rceil$ iterations with a convex combination that is at most $\delta$ below the optimum value $g$.*

*Proof.* We begin by observing that if the optimization problem at iteration $t$ is infeasible, then $\gamma_t^*(\nu) > \widehat{\gamma}_t - \delta \geq g - \delta$. Also if $\mathbf{d}^t$ contains a zero, then since the objective function $\Delta(\mathbf{d}, \mathbf{d}^0)$ is strictly convex in $\mathbf{d}$ and minimized at the interior point $\mathbf{d}^0$, there is no optimal solution in the interior of the simplex. Hence, $\gamma_t^*(\nu) = \widehat{\gamma}_t - \delta \geq g - \delta$.

Let $\mathcal{C}_t$ be the convex subset of probability vectors $\mathbf{d} \in \mathcal{P}^N$ satisfying $\mathbf{d} \leq \frac{1}{\nu}\mathbf{1}$ and $\max_{m=1}^t \mathbf{d} \cdot \mathbf{u}^t \leq \widehat{\gamma}_t - \delta$. Notice that $\mathcal{C}_0$ is the $N$ dimensional probability simplex where the components are capped to $\frac{1}{\nu}$. The distribution $\mathbf{d}^{t-1}$ at iteration $t - 1$ is the projection of $\mathbf{d}^0$ onto the closed convex set $\mathcal{C}_{t-1}$. Because adding a new hypothesis in iteration $t$ results in an additional constraint and $\widehat{\gamma}_t \leq \widehat{\gamma}_{t-1}$,

we have $\mathcal{C}_t \subseteq \mathcal{C}_{t-1}$. If $t \leq T-1$, then our termination condition assures that at iteration $t-1$ the set $\mathcal{C}_{t-1}$ has a feasible solution in the interior of the simplex. Also, $\mathbf{d}^0$ lies in the interior and $\mathbf{d}^t \in \mathcal{C}_t \subseteq \mathcal{C}_{t-1}$. These preconditions assure that at iteration $t-1$, the projection $\mathbf{d}^{t-1}$ of $\mathbf{d}^0$ onto $\mathcal{C}_{t-1}$, exists and the Generalized Pythagorean Theorem for Bregman divergences [2, 9] is applicable:

$$\Delta(\mathbf{d}^t, \mathbf{d}^0) - \Delta(\mathbf{d}^{t-1}, \mathbf{d}^0) \geq \Delta(\mathbf{d}^t, \mathbf{d}^{t-1}). \tag{3}$$

By Pinsker's inequality, $\Delta(\mathbf{d}^t, \mathbf{d}^{t-1}) \geq \frac{(||\mathbf{d}^t - \mathbf{d}^{t-1}||_1)^2}{2}$, and by Hölder's inequality, $||\mathbf{d}^{t-1} - \mathbf{d}^t||_1 \geq ||\mathbf{d}^{t-1} - \mathbf{d}^t||_1 ||\mathbf{u}^t||_\infty \geq \mathbf{d}^{t-1} \cdot \mathbf{u}^t - \mathbf{d}^t \cdot \mathbf{u}^t$. Also $\mathbf{d}^{t-1} \cdot \mathbf{u}^t \geq \widehat{\gamma}_t$ by the definition of $\widehat{\gamma}_t$, and the constraints on the optimization problem assure that $\mathbf{d}^t \cdot \mathbf{u}^t \leq \widehat{\gamma}_t - \delta$ and thus $\mathbf{d}^{t-1} \cdot \mathbf{u}^t - \mathbf{d}^t \cdot \mathbf{u}^t \geq \widehat{\gamma}_t - (\widehat{\gamma}_t - \delta) = \delta$. We conclude that $\Delta(\mathbf{d}^t, \mathbf{d}^{t-1}) \geq \frac{\delta^2}{2}$ at iterations 1 through $T-1$. By summing (3) over the first $T-1$ iterations, we obtain

$$\Delta(\mathbf{d}^T, \mathbf{d}^0) - \Delta(\mathbf{d}^0, \mathbf{d}^0) \geq (T-1)\frac{\delta^2}{2}.$$

Since the left side is at most $\ln(N/\nu)$, the bound of the theorem follows. □

When $\nu = 1$, then capping is vacuous and the algorithm and its iteration bound coincides with the bound for TotalBoost. Note that the upper bound $\ln(N/\nu)$ on the relative entropy decreases with $\nu$. When $\nu = N$, then the distribution stays at $\mathbf{d}^0$ and the iteration bound is zero.

## 5 Experiments

In a first study, we use experiments on synthetic data to illustrate the general behavior of the considered algorithms.[2] We generated a synthetic data set by starting with a random matrix of 2000 rows and 100 columns, where each entry was chosen uniformly in $[0, 1]$. For the first 1000 rows, we added $1/2$ to the first 10 columns and rescaled such that the entries in those columns were again in $[0, 1]$. The rows of this matrix are our examples and the columns and their negation are the base hypotheses, giving us a total of 200 of them. The first 1000 examples were labeled $+1$ and the rest $-1$. This results in a well separable dataset. To illustrate how the algorithms deal with the inseparable case, we flipped the sign of a random $10\%$ of the data set. We then chose a random 500 examples as our training set and the rest as our test set. In every boosting iteration we chose the base hypothesis which has the largest edge with respect to the current distribution on the examples.

We have trained LPBoost and SoftBoost for different values of $\nu$ and recorded the generalization error (cf. Figure 2; $\delta = 10^{-3}$). We should expect that for small $\nu$ (e.g. $\nu/N < 10\%$) the data is not easily separable, even when allowing $\nu$ wrong predictions. Hence the algorithm may mistakenly concentrate on the random directions for discrimination. If $\nu$ is large enough, most incorrectly labeled examples are likely to be identified as margin errors ($\psi_i > 0$) and the performance should stabilize. In Figure 2 we observe this expected behavior and also that for large $\nu$ the classification performance decays again. The generalization performances of LPBoost and SoftBoost are very similar, which is expected as they both attempt to maximize the soft-margin.

Using the same data set, we analysed the convergence speed of several algorithms: LPBoost, SoftBoost, BrownBoost, and SmoothBoost. We chose $\delta = 10^{-2}$ and $\nu = 200$.[3] For every iteration we record all margins and compute the soft margin objective (1) for optimally chosen $\rho$ and $\psi$'s. Figure 3 plots this value against the number of iterations for the four algorithms. SmoothBoost takes dramatically longer to converge to the maximum soft margin than the other other three algorithms. In our experiments it nearly converges to the maximum soft margin objective, even though no theoretical evidence is known for this observed convergence. Among the three remaining algorithms, LPBoost and SoftBoost converge in roughly the same number of iterations, but SoftBoost has a slower start. BrownBoost terminates in fewer iterations than the other algorithms but does not maximize the soft margin.[4] This is not surprising as there is no theoretical reason to expect such a result.

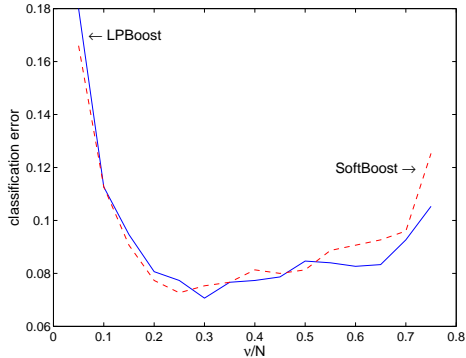

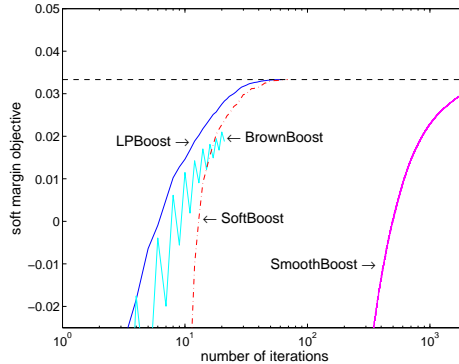

Figure 2: Generalization performance of SoftBoost (solid) and LPBoost (dotted) on a synthetic data set with 10% label-noise for different values of $\nu$.

Figure 3: Soft margin objective vs. the number of iterations for LPBoost, SoftBoost, BrownBoost and SmoothBoost.

Finally, we present a small comparison on ten benchmark data sets derived from the UCI benchmark repository as previously used in [15]. We analyze the performance of AdaBoost, LPBoost, Soft-Boost, BrownBoost [6] and AdaBoost$_{Reg}$ [15] using RBF networks as base learning algorithm.[5] The data comes in 100 predefined splits into training and test sets. For each of the splits we use 5-fold cross-validation to select the optimal regularization parameter for each of the algorithms. This leads to 100 estimates of the generalization error for each method and data set. The means and standard deviations are given in Table 1.[6] As before, the generalization performances of Soft-Boost and LPBoost are very similar. However, the soft margin algorithms outperform AdaBoost on most data sets. The genaralization error of BrownBoost lies between that of AdaBoost and Soft-Boost. AdaBoost$_{Reg}$ performs as well as SoftBoost, but there are no iteration bounds known for this algorithm.

Even though SoftBoost and LPBoost often have similar generalization error on natural datasets, the number of iterations needed by both algorithms can be radically different (see Theorem 1). Also, in [22] there are some artificial data sets where TotalBoost (i.e. SoftBoost with $\nu = 1$) outperformed LPBoost i.t.o. generalization error.

| | AdaBoost | | | LPBoost | | | SoftBoost | | | BrownBoost | | | AdaBoost reg | | |
|---|---|---|---|---|---|---|---|---|---|---|---|---|---|---|---|
| Banana | 13.3 | $\pm$ | 0.7 | 11.1 | $\pm$ | 0.6 | 11.1 | $\pm$ | 0.5 | 12.9 | $\pm$ | 0.7 | 11.3 | $\pm$ | 0.6 |
| B.Cancer | 32.1 | $\pm$ | 3.8 | 27.8 | $\pm$ | 4.3 | 28.0 | $\pm$ | 4.5 | 30.2 | $\pm$ | 3.9 | 27.3 | $\pm$ | 4.3 |
| Diabetes | 27.9 | $\pm$ | 1.5 | 24.4 | $\pm$ | 1.7 | 24.4 | $\pm$ | 1.7 | 27.2 | $\pm$ | 1.6 | 24.5 | $\pm$ | 1.7 |
| German | 26.9 | $\pm$ | 1.9 | 24.6 | $\pm$ | 2.1 | 24.7 | $\pm$ | 2.1 | 24.8 | $\pm$ | 1.9 | 25.0 | $\pm$ | 2.2 |
| Heart | 20.1 | $\pm$ | 2.7 | 18.4 | $\pm$ | 3.0 | 18.2 | $\pm$ | 2.7 | 20.0 | $\pm$ | 2.8 | 17.6 | $\pm$ | 3.0 |
| Ringnorm | 1.9 | $\pm$ | 0.3* | 1.9 | $\pm$ | 0.2 | 1.8 | $\pm$ | 0.2 | 1.9 | $\pm$ | 0.2 | 1.7 | $\pm$ | 0.2 |
| F.Solar | 36.1 | $\pm$ | 1.5 | 35.7 | $\pm$ | 1.6 | 35.5 | $\pm$ | 1.4 | 36.1 | $\pm$ | 1.4 | 34.4 | $\pm$ | 1.7 |
| Thyroid | 4.4 | $\pm$ | 1.9* | 4.9 | $\pm$ | 1.9 | 4.9 | $\pm$ | 1.9 | 4.6 | $\pm$ | 2.1 | 4.9 | $\pm$ | 2.0 |
| Titanic | 22.8 | $\pm$ | 1.0 | 22.8 | $\pm$ | 1.0 | 23.0 | $\pm$ | 0.8 | 22.8 | $\pm$ | 0.8 | 22.7 | $\pm$ | 1.0 |
| Waveform | 10.5 | $\pm$ | 0.4 | 10.1 | $\pm$ | 0.5 | 9.8 | $\pm$ | 0.5 | 10.4 | $\pm$ | 0.4 | 10.4 | $\pm$ | 0.7 |

Table 1: Generalization error estimates and standard deviations for ten UCI benchmark data sets. SoftBoost and LPBoost outperform AdaBoost and BrownBoost on most data sets.

## 6   Conclusion

We prove by counterexample that LPBoost cannot have an $O(\ln N)$ iteration bound. This counterexample may seem similar to the proof that the Simplex algorithm for LP can take exponentially more steps than interior point methods. However this similarity is only superficial. First, our iteration bound does not depend on the LP solver used within LPBoost. This is because in the construction, the interim solutions are always unique and thus all LP solvers will produce the same solution. Second, the iteration bound essentially says that column generation methods (of which LPBoost is a canonical example) should not solve the current subproblem at iteration $t$ optimally. Instead a good algorithm should *loosen the constraints and spread the weight* via a regularization such as the relative entropy. These two tricks used by the SoftBoost algorithm make it possible to obtain iteration

bounds that grow logarithmic in $N$. The iteration bound for our algorithm is a straightforward extension of a bound given in [22] that is based on Bregman projection methods. By using a different divergence in SoftBoost, such as the sum of binary relative entropies, the algorithm morphs into a "soft" version of LogitBoost (see discussion in [22]) which has essentially the same iteration bound as SoftBoost. We think that the use of Bregman projections illustrates the generality of the methods. Although the proofs seem trivial in hindsight, simple logarithmic iteration bounds for boosting algorithms that maximize the soft margin have eluded many researchers (including the authors) for a long time. Note that duality methods typically can be used in place of Bregman projections. For example in [12], a number of iteration bounds for boosting algorithms are proven with both methods.

On a more technical level, we show that LPBoost may require $N/2$ examples to get .99 close to the maximum hard margin. We believe that similar methods can be used to show that $\Omega(N/\delta)$ examples may be needed to get $\delta$ close. However the real challenge is to prove that LPBoost may require $\Omega(N/\delta^2)$ examples to get $\delta$ close.

## Footnotes

[1] Please note that [20] have previously used the parameter $\nu$ with a slightly different meaning, namely $\nu/N$ in our notation. We use an unnormalized version of $\nu$ denoting a *number* of examples instead of a fraction.

[2]Our code is available at `https://sourceforge.net/projects/nboost`

[3]Smaller choices of $\nu$ lead to an even slower convergence of SmoothBoost.

[4]SmoothBoost has two parameters: a guarantee $g$ on the edge of the base learner and the target margin $\theta$. We chose $g = \gamma^*(\nu)$ (computed with LPBoost) and $\theta = \frac{g/2}{2+g/2}$ as proposed in [21]. Brownboost's one parameter, $c = 0.35$, was chosen via cross-validation.

[5]The data is from `http://theoval.cmp.uea.ac.uk/~gcc/matlab/index.shtml`. The RBF networks were obtained from the authors of [15], including the hyper-parameter settings for each data set.

[6]Note that [15] contains a similar benchmark comparison. It is based on a different model selection setup leading to underestimates of the generalization error. Presumably due to slight differences in the RBF hyper-parameters settings, our results for AdaBoost often deviate by 1-2%.

## References

[1] L. Breiman. Prediction games and arcing algorithms. *Neural Computation*, 11(7):1493–1518, 1999. Also Technical Report 504, Statistics Department, University of California Berkeley.

[2] Y. Censor and S. A. Zenios. *Parallel Optimization*. Oxford, New York, 1997.

[3] C. Cortes and V. Vapnik. Support-vector networks. *Machine Learning*, 20(3):273–297, 1995.

[4] A. Demiriz, K.P. Bennett, and J. Shawe-Taylor. Linear programming boosting via column generation. *Machine Learning*, 46(1-3):225–254, 2002.

[5] C. Domingo and O. Watanabe. Madaboost: A modification of Adaboost. In *Proc. COLT '00*, pages 180–189, 2000.

[6] Y. Freund. An adaptive version of the boost by majority algorithm. *Mach. Learn.*, 43(3):293–318, 2001.

[7] Y. Freund and R.E. Schapire. A decision-theoretic generalization of on-line learning and an application to boosting. *Journal of Computer and System Sciences*, 55(1):119–139, 1997.

[8] A.J. Grove and D. Schuurmans. Boosting in the limit: Maximizing the margin of learned ensembles. In *Proceedings of the Fifteenth National Conference on Artifical Intelligence*, 1998.

[9] Mark Herbster and Manfred K. Warmuth. Tracking the best linear predictor. *Journal of Machine Learning Research*, 1:281–309, 2001.

[10] R. Hettich and K.O. Kortanek. Semi-infinite programming: Theory, methods and applications. *SIAM Review*, 3:380–429, September 1993.

[11] J. Kivinen and M. K. Warmuth. Boosting as entropy projection. In *Proc. 12th Annu. Conference on Comput. Learning Theory*, pages 134–144. ACM Press, New York, NY, 1999.

[12] J. Liao. *Totally Corrective Boosting Algorithms that Maximize the Margin*. PhD thesis, University of California at Santa Cruz, December 2006.

[13] R. Meir and G. Rätsch. An introduction to boosting and leveraging. In S. Mendelson and A. Smola, editors, *Proc. 1st Machine Learning Summer School, Canberra*, LNCS, pages 119–184. Springer, 2003.

[14] G. Rätsch. *Robust Boosting via Convex Optimization: Theory and Applications*. PhD thesis, University of Potsdam, Germany, December 2001.

[15] G. Rätsch, T. Onoda, and K.-R. Müller. Soft margins for AdaBoost. *Machine Learning*, 42(3):287–320, 2001.

[16] G. Rätsch, B. Schölkopf, A.J. Smola, S. Mika, T. Onoda, and K.-R. Müller. Robust ensemble learning. In A.J. Smola, P.L. Bartlett, B. Schölkopf, and D. Schuurmans, editors, *Advances in Large Margin Classifiers*, pages 207–219. MIT Press, Cambridge, MA, 2000.

[17] G. Rätsch and M. K. Warmuth. Efficient margin maximizing with boosting. *Journal of Machine Learning Research*, 6:2131–2152, December 2005.

[18] C. Rudin, I. Daubechies, and R.E. Schapire. The dynamics of adaboost: Cyclic behavior and convergence of margins. *Journal of Machine Learning Research*, 5:1557–1595, 2004.

[19] R.E. Schapire, Y. Freund, P.L. Bartlett, and W.S. Lee. Boosting the margin: A new explanation for the effectiveness of voting methods. *The Annals of Statistics*, 26(5):1651–1686, 1998.

[20] B. Schölkopf, A.J. Smola, R.C. Williamson, and P.L. Bartlett. New support vector algorithms. *Neural Comput.*, 12(5):1207–1245, 2000.

[21] Rocco A. Servedio. Smooth boosting and learning with malicious noise. *Journal of Machine Learning Research*, 4:633–648, 2003.

[22] M.K. Warmuth, J. Liao, and G. Rätsch. Totally corrective boosting algorithms that maximize the margin. In *Proc. ICML '06*, pages 1001–1008. ACM Press, 2006.

